# Graphical Models for Recognizing Human Interactions

**Nuria M. Oliver, Barbara Rosario and Alex Pentland**
20 Ames Street, E15-384C,
Media Arts and Sciences Laboratory, MIT
Cambridge, MA 02139
{nuria, rosario, sandy}@media.mit.edu

## Abstract

We describe a real-time computer vision and machine learning system for modeling and recognizing human actions and interactions. Two different domains are explored: recognition of two-handed motions in the martial art 'Tai Chi', and multiple-person interactions in a visual surveillance task. Our system combines top-down with bottom-up information using a feedback loop, and is formulated with a Bayesian framework. Two different graphical models (HMMs and Coupled HMMs) are used for modeling both individual actions and multiple-agent interactions, and CHMMs are shown to work more efficiently and accurately for a given amount of training. Finally, to overcome the limited amounts of training data, we demonstrate that 'synthetic agents' (Alife-style agents) can be used to develop flexible prior models of the person-to-person interactions.

## 1 INTRODUCTION

We describe a real-time computer vision and machine learning system for modeling and recognizing human behaviors in two different scenarios: (1) complex, two-handed action recognition in the martial art of *Tai Chi* and (2) detection and recognition of individual human behaviors and multiple-person interactions in a visual surveillance task. In the latter case, the system is particularly concerned with detecting when interactions between people occur, and classifying them.

Graphical models, such as Hidden Markov Models (HMMs) [6] and Coupled Hidden Markov Models (CHMMs) [3, 2], seem appropriate for modeling and classifying human behaviors because they offer dynamic time warping, a well-understood training algorithm, and a clear Bayesian semantics for both individual (HMMs) and interacting or coupled (CHMMs) generative processes. A major problem with this data-driven statistical approach, especially when modeling rare or anomalous behaviors, is the limited number of training examples. A major emphasis of our work, therefore, is on efficient Bayesian integration of both prior knowledge with evidence from data. We will show that for situations involving multiple independent (or partially independent) agents the Coupled HMM approach generates much better results than traditional HMM methods.

In addition, we have developed a *synthetic agent or Alife* modeling environment for building and training flexible a *priori* models of various behaviors using software agents. Simulation with these software agents yields synthetic data that can be used to train prior models. These prior models can then be used recursively in a Bayesian framework to fit real behavioral data.

This synthetic agent approach is a straightforward and flexible method for developing prior models, one that does not require strong analytical assumptions to be made about the form of the priors[1]. In addition, it has allowed us to develop robust models even when there are only a few examples of some target behaviors. In our experiments we have found that by combining such synthetic priors with limited real data we can easily achieve very high accuracies at recognition of different human-to-human interactions.

The paper is structured as follows: section 2 presents an overview of the system, the statistical models used for behavior modeling and recognition are described in section 3. Section 4 contains experimental results in two different real situations. Finally section 5 summarizes the main conclusions and our future lines of research.

## 2   VISUAL INPUT

We have experimented using two different types of visual input. The first is a real-time, self-calibrating 3-D stereo blob tracker (used for the *Tai Chi* scenario) [1], and the second is a real-time blob-tracking system [5] (used in the visual surveillance task). In both cases an Extended Kalman filter (EKF) tracks the blobs' location, coarse shape, color pattern, and velocity. This information is represented as a low-dimensional, parametric probability distribution function (PDF) composed of a mixture of Gaussians, whose parameters (sufficient statistics and mixing weights for each of the components) are estimated using Expectation Maximization (EM).

This visual input module detects and tracks moving objects — body parts in *Tai Chi* and pedestrians in the visual surveillance task — and outputs a feature vector describing their motion, heading, and spatial relationship to all nearby moving objects. These output feature vectors constitute the temporally ordered stream of data input to our stochastic state-based behavior models. Both HMMs and CHMMs, with varying structures depending on the complexity of the behavior, are used for classifying the observed behaviors.

Both *top-down* and *bottom-up* flows of information are continuously managed and combined for each moving object within the scene. The Bayesian graphical models offer a mathematical framework for combining the observations (bottom-up) with complex behavioral priors (top-down) to provide expectations that will be fed back to the input visual system.

## 3   VISUAL UNDERSTANDING VIA GRAPHICAL MODELS: HMMs and CHMMs

Statistical directed acyclic graphs (DAGs) or probabilistic inference networks (PINs hereafter) can provide a computationally efficient solution to the problem of time series analysis and modeling. HMMs and some of their extensions, in particular CHMMs, can be viewed as a particular and simple case of temporal PIN or DAG. Graphically Markov Models are often depicted 'rolled-out in time' as Probabilistic Inference Networks, such as in figure 1. PINs present important advantages that are relevant to our problem: they can handle incomplete data as well as uncertainty; they are trainable and easier to avoid overfitting; they encode causality in a natural way; there are algorithms for both doing prediction and probabilistic inference; they offer a framework for combining prior knowledge and data; and finally they are modular and parallelizable.

Traditional HMMs offer a probabilistic framework for modeling processes that have structure in time. They offer clear Bayesian semantics, efficient algorithms for state and parameter estimation, and they automatically perform dynamic time warping. An HMM is essentially a quantization of a system's configuration space into a small number of discrete states, together with probabilities for transitions between

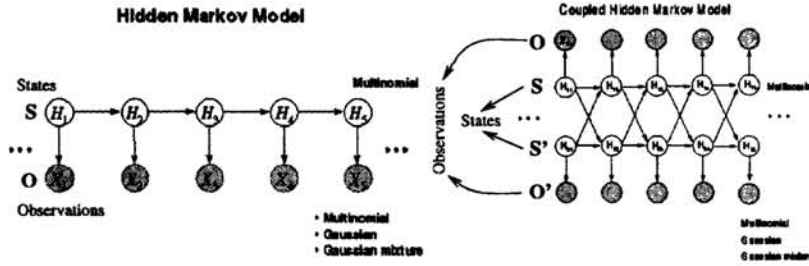

Figure 1: Graphical representation of a HMM and a CHMM rolled-out in time

states. A single finite discrete variable indexes the current state of the system. Any information about the history of the process needed for future inferences must be reflected in the current value of this state variable.

However many interesting real-life problems are composed of multiple interacting processes, and thus merit a compositional representation of two or more variables. This is typically the case for systems that have structure both in time and space. With a single state variable, Markov models are ill-suited to these problems. In order to model these interactions a more complex architecture is needed.

Extensions to the basic Markov model generally increase the memory of the system (durational modeling), providing it with compositional state in time. We are interested in systems that have compositional state in *space*, e.g., more than one simultaneous state variable. It is well known that the exact solution of extensions of the basic HMM to 3 or more chains is intractable. In those cases approximation techniques are needed ([7, 4, 8, 9]). However, it is also known that there exists an exact solution for the case of 2 interacting chains, as it is our case [7, 2].

We therefore use two Coupled Hidden Markov Models (CHMMs) for modeling two interacting processes, whether they are separate body parts or individual humans. In this architecture state chains are coupled via matrices of conditional probabilities modeling causal (temporal) influences between their hidden state variables. The graphical representation of CHMMs is shown in figure 1. From the graph it can be seen that for each chain, the state at time $t$ depends on the state at time $t-1$ in both chains. The influence of one chain on the other is through a causal link.

In this paper we compare performance of HMMs and CHMMs for maximum *a posteriori* (MAP) state estimation. We compute the most likely sequence of states $\hat{S}$ within a model given the observation sequence $O = \{o_1, \ldots, o_n\}$. This most likely sequence is obtained by $\hat{S} = argmax_S P(S|O)$.

In the case of HMMs the posterior state sequence probability $P(S|O)$ is given by

$$P(S|O) = P_{s_1} p_{s_1}(o_1) \prod_{t=2}^{T} p_{s_t}(o_t) P_{s_t|s_{t-1}} \tag{1}$$

where $S = \{a_1, \ldots, a_N\}$ is the set of discrete states, $s_t \in S$ corresponds to the state at time $t$. $P_{i|j} \doteq P_{s_t=a_i|s_{t-1}=a_j}$ is the state-to-state transition probability (i.e. probability of being in state $a_i$ at time $t$ given that the system was in state $a_j$ at time $t-1$). In the following we will write them as $P_{s_t|s_{t-1}}$. $P_i \doteq P_{s_1=a_i} = P_{s_1}$ are the prior probabilities for the initial state. Finally $p_i(o_t) \doteq p_{s_t=a_i}(o_t) = p_{s_t}(o_t)$ are the output probabilities for each state[2].

For CHMMs we need to introduce another set of probabilities, $P_{s_t|s'_{t-1}}$, which cor-

respond to the probability of state $s_t$ at time $t$ in one chain given that the other chain –denoted hereafter by superscript $'$ – was in state $s'_{t-1}$ at time $t-1$. These new probabilities express the causal influence (coupling) of one chain to the other. The posterior state probability for CHMMs is expressed as

$$P(S|O) = \frac{P_{s_1} p_{s_1}(o_1) P_{s'_1} p_{s'_1}(o'_1)}{P(O)} \times \prod_{t=2}^{T} P_{s_t|s_{t-1}} P_{s'_t|s'_{t-1}} P_{s'_t|s_{t-1}} P_{s_t|s'_{t-1}} p_{s_t}(o_t) p_{s'_t}(o'_t)$$

(2)

where $s_t, s'_t; o_t, o'_t$ denote states and observations for each of the Markov chains that compose the CHMMs.

In [2] a deterministic approximation for maximum *a posterior* (MAP) state estimation is introduced. It enables fast classification and parameter estimation via EM, and also obtains an upper bound on the cross entropy with the full (combinatoric) posterior which can be minimized using a subspace that is linear in the number of state variables. An "N-heads" dynamic programming algorithm samples from the $O(N)$ highest probability paths through a compacted state trellis, with complexity $O(T(CN)^2)$ for $C$ chains of $N$ states apiece observing $T$ data points. The cartesian product equivalent HMM would involve a combinatoric number of states, typically requiring $O(TN^{2C})$ computations. We are particularly interested in efficient, compact algorithms that can perform in real-time.

## 4   EXPERIMENTAL RESULTS

Our first experiment is with a version of Tai Chi Ch'uan (a Chinese martial and meditative art) that is practiced while sitting. Using our self-calibrating, 3-D stereo blob tracker [1], we obtained 3D hand tracking data for three Tai Chi gestures involving two, semi-independent arm motions: the left single whip, the left cobra, and the left brush knee. Figure 4 illustrates one of the gestures and the blob-tracking. A detailed description of this set of *Tai Chi* experimental results can be found in [3] and viewed at `http://nuria.www.media.mit.edu/~nuria/chmm/taichi.html`.

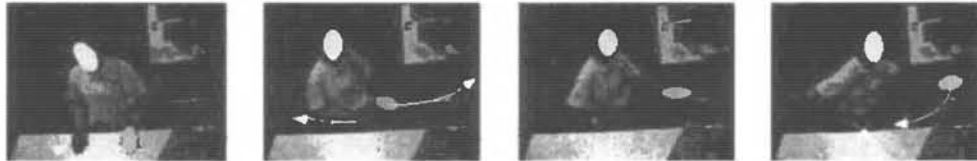

Figure 2: Selected frames from 'left brush knee.'

We collected 52 sequences, roughly 17 of each gesture and created a feature vector consisting of the 3-D $(x, y, z)$ centroid (mean position) of each of the blobs that characterize the hands. The resulting six-dimensional time series was used for training both HMMs and CHMMs.

We used the best trained HMMs and CHMMs — using 10-crossvalidation — to classify the full data set of 52 gestures. The Viterbi algorithm was used to find the maximum likelihood model for HMMs and CHMMs. Two-thirds of the testing data had not been seen in training, including gestures performed at varying speeds and from slightly different views. It can be seen from the classification accuracies, shown in table 1, that the CHMMs outperform the HMMs. This difference is not due to intrinsic modeling power, however; from earlier experiments we know that when a large number of training samples is available then HMMs can reach similar accuracies. We conclude thus that for data where there are two partially-independent processes (e.g., coordinated but not exactly linked), the CHMM method requires much less training to achieve a high classification accuracy.

Table 1 illustrates the source of this training advantage. The numbers between

Table 1: Recognition accuracies for HMMs and CHMMs on Tai Chi gestures. The expressions between parenthesis correspond to the number of parameters of the largest best-scoring model.

| Recognition Results on Tai Chi Gestures | | |
| --- | --- | --- |
| | Single HMMs | Coupled HMMs (CHMMs) |
| Accuracy | 69.2% (25+30+180) | 100% (27+18+54) |

parenthesis correspond to the number of degrees of freedom in the largest best-scoring model: state-to-state probabilities + output means + output covariances. The conventional HMM has a large number of covariance parameters because it has a 6-D output variable; whereas the CHMM architecture has two 3-D output variables. In consequence, due to their larger dimensionality HMMs need much more training data than equivalent CHMMs before yielding good generalization results.

Our second experiment was with a pedestrian video surveillance task [3]; the goal was first to recognize typical pedestrian behaviors in an open plaza (e.g., walk from A to B, run from C to D), and second to recognize interactions between the pedestrians (e.g., person X greets person Y). The task is to reliably and robustly detect and track the pedestrians in the scene. We use in this case 2-D *blob features* for modeling each pedestrian. In our system one of the main cues for clustering the pixels into blobs is motion, because we have a static background with moving objects. To detect these moving objects we build an eigenspace that models the background. Depending on the dynamics of the background scene the system can *adaptively* relearn the eigenbackground to compensate for changes such as big shadows.

The trajectories of each blob are computed and saved into a *dynamic track memory*. Each trajectory has associated a first order EKF that predicts the blob's position and velocity in the next frame As before, the appearance of each blob is modeled by a Gaussian PDF in RGB color space, allowing us to handle occlusions.

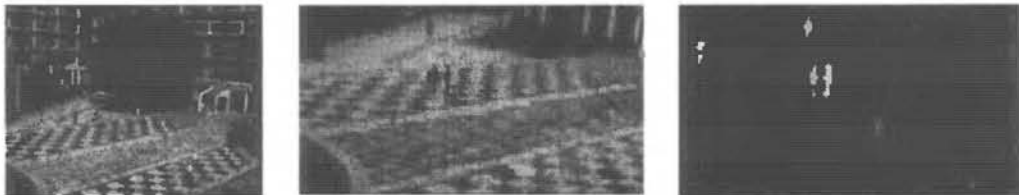

Figure 3: Typical Image from pedestrian plaza. Background mean image, input image with blob bounding boxes and blob segmentation image

The behaviors we examine are generated by pedestrians walking in an open outdoor environment. Our goal is to develop a generic, compositional analysis of the observed behaviors in terms of states and transitions between states over time in such a manner that (1) the states correspond to our common sense notions of human behaviors, and (2) they are immediately applicable to a wide range of sites and viewing situations. Figure 3 shows a typical image for our pedestrian scenario, the pedestrians found, and the final segmentation. Two people (each modeled as its own generative process) may interact without wholly determining each others' behavior. Instead, each of them has its own internal dynamics and is influenced (either weakly or strongly) by others. The probabilities $P_{s_t|s'_{t-1}}$ and $P_{s'_t|s_{t-1}}$ from equation 2 describe this kind of interactions and CHMMs are intended to model them in as efficient a manner as is possible.

We would like to have a system that will accurately interpret behaviors and interactions within almost any pedestrian scene with at most minimal training. As we have

http://www.vismod.www.media.mit.edu/ nuria/humanBehavior/humanBehavior.html

already mentioned, one critical problem is the generation of models that capture our prior knowledge about human behavior. To achieve this goal we have developed a modeling environment that uses synthetic agents to mimic pedestrian behavior in a virtual environment. The agents can be assigned different behaviors and they can interact with each other as well. Currently they can generate 5 different interacting behaviors and various kinds of individual behaviors (with no interaction). These behaviors are: following, meet and walk together (inter1); approach, meet and go on separately (inter2) or go on together (inter3); change direction in order to meet, approach, meet and continue together (inter4) or go on separately (inter5). The parameters of this virtual environment are modeled using data drawn from a 'generic' set of real scenes.

By training the models of the synthetic agents to have good generalization and invariance properties, we can obtain flexible prior models for use when learning the human behavior models from real scenes. Thus the synthetic prior models allow us to learn robust behavior models from a small number of real behavior examples. This capability is of special importance in a visual surveillance task, where typically the behaviors of greatest interest are also the rarest.

To test our behavior modeling in the pedestrian scenario, we first used the detection and tracking system previously described to obtain 2-D blob features for each person in several hours of video. More than 20 examples of *following* and the two first types of *meeting* behaviors were detected and processed.

CHMMs were then used for modeling three different behaviors: following, meet and continue together, and meet and go on separately. Furthermore, an *interaction* versus *no interaction* detection test was also performed (HMMs performed so poorly at this task that their results are not reported). In addition to velocity, heading, and position, the feature vectors consisted of the derivative of the relative distance between two agents, their degree of alignment (dot product of their velocity vectors) and the magnitude of the difference in their velocity vectors.

We tested on this video data using models trained with two types of data: (1) 'Prior-only models', that is, models learned entirely from our synthetic-agents environment and then applied directly to the real data with no additional training or tuning of the parameters; and (2) 'Posterior models', or prior-plus-real data behavior models trained by starting with the prior-only model and then 'tuning' the models with data from this specific site, using eight examples of each type of interaction. Recognition accuracies for both these 'prior' and 'posterior' CHMMs are summarized in table 2. It is noteworthy that with only 8 training examples, the recognition accuracy on the remaining data could be raised to 100%. This demonstrates the ability to accomplish extremely rapid refinement of our behavior models from the initial a priori models.

Table 2: Accuracies on real pedestrian data, (a) only a priori models, (b) posterior models (with site-specific training)

| Accuracy on Real Pedestrian Data | | | | |
|---|---|---|---|---|
| | No-inter | Inter1 | Inter2 | Inter3 |
| (a)**Prior CHMMs** | 90.9 | 93.7 | 100 | 100 |
| (b)**Posterior CHMMs** | 100 | 100 | 100 | 100 |

In a visual surveillance system the *false alarm* rate is often as important as the classification accuracy[4] To analyze this aspect of our system's performance, we calculated the system's ROC curve. For accuracies of 95% the false alarm rate was less than 0.01.

# 5   SUMMARY, CONCLUSIONS AND FUTURE WORK

In this paper we have described a computer vision system and a mathematical modeling framework for recognizing different human behaviors and interactions in two different real domains: human actions in the martial art of *Tai Chi* and human interactions in a visual surveillance task. Our system combines top-down with bottom-up information in a closed feedback loop, with both components employing a statistical Bayesian approach.

Two different state-based statistical learning architectures, namely HMMs and CHMMs, have been proposed and compared for modeling behaviors and interactions. The superiority of the CHMM formulation has been demonstrated in terms of both training efficiency and classification accuracy. A synthetic agent training system has been created in order to develop flexible prior behavior models, and we have demonstrated the ability to use these prior models to accurately classify real behaviors with no additional training on real data. This fact is specially important, given the limited amount of training data available.

Future directions under current investigation include: extending our agent interactions to more than two interacting processes; developing a hierarchical system where complex behaviors are expressed in terms of simpler behaviors; automatic discovery and modeling of new behaviors (both structure and parameters); automatic determination of priors, their evaluation and interpretation; developing an attentional mechanism with a foveated camera along with a more detailed representation of the behaviors; evaluating the adaptability of off-line learned behavior structures to different real situations; and exploring a sampling approach for recognizing behaviors by sampling the interactions generated by our synthetic agents.

## Acknowledgments

Sincere thanks to Michael Jordan, Tony Jebara and Matthew Brand for their inestimable help.

## Footnotes

[1]Note that our priors have the same form as our posteriors, namely, they are graphical models.

[2]The output probability is the probability of observing $o_t$ given state $a_t$ at time $t$

[3]    Further    information    about    this    system    can    be    found    at

[4]In an ideal automatic surveillance system, all the targeted behaviors should be detected with a close-to-zero false alarm rate, so that we can reasonably alert a human operator to examine them further.

## References

1. A. Azarbayejani and A. Pentland.   Real-time self-calibrating stereo person-tracker using 3-D shape estimation from blob features.   In *Proceedings, International Conference on Pattern Recognition*, Vienna, August 1996. IEEE.

2. M. Brand.   Coupled hidden markov models for modeling interacting processes. November 1996.   Submitted to Neural Computation.

3. M. Brand and N. Oliver.   Coupled hidden markov models for complex action recognition.   In *In Proceedings of IEEE CVPR97*, 1996.

4. Z. Ghahramani and M. I. Jordan.   Factorial hidden Markov models.   In D. S. Touretzky, M. C. Mozer, and M. Hasselmo, editors, *NIPS*, volume 8, Cambridge, MA, 1996. MITP.

5. N. Oliver, B. Rosario, and A. Pentland.   Statistical modeling of human behaviors. In *To appear in Proceedings of CVPR98, Perception of Action Workshop*, 1998.

6  L. R. Rabiner.   A tutorial on hidden markov models and selected applications in speech recognition.   *PIEEE*, 77(2):257–285, 1989.

7. L. K. Saul and M. I. Jordan.   Boltzmann chains and hidden Markov models.   In G. Tesauro, D. S. Touretzky, and T. Leen, editors, *NIPS*, volume 7, Cambridge, MA, 1995. MITP.

8. P. Smyth, D. Heckerman, and M. Jordan.   Probabilistic independence networks for hidden Markov probability models.   AI memo 1565, MIT, Cambridge, MA, Feb 1996.

9  C. Williams and G. E. Hinton.   Mean field networks that learn to discriminate temporally distorted strings.   In *Proceedings, connectionist models summer school*, pages 18–22, San Mateo, CA, 1990. Morgan Kaufmann.
